# Estimating disparity with confidence from energy neurons

**Eric K. C. Tsang**
Dept. of Electronic and Computer Engr.
Hong Kong Univ. of Sci. and Tech.
Kowloon, HONG KONG SAR
eeeric@ee.ust.hk

**Bertram E. Shi**
Dept. of Electronic and Computer Engr.
Hong Kong Univ. of Sci. and Tech.
Kowloon, HONG KONG SAR
eebert@ee.ust.hk

## Abstract

The peak location in a population of phase-tuned neurons has been shown to be a more reliable estimator for disparity than the peak location in a population of position-tuned neurons. Unfortunately, the disparity range covered by a phase-tuned population is limited by phase wraparound. Thus, a single population cannot cover the large range of disparities encountered in natural scenes unless the scale of the receptive fields is chosen to be very large, which results in very low resolution depth estimates. Here we describe a biologically plausible measure of the confidence that the stimulus disparity is inside the range covered by a population of phase-tuned neurons. Based upon this confidence measure, we propose an algorithm for disparity estimation that uses many populations of high-resolution phase-tuned neurons that are biased to different disparity ranges via position shifts between the left and right eye receptive fields. The population with the highest confidence is used to estimate the stimulus disparity. We show that this algorithm outperforms a previously proposed coarse-to-fine algorithm for disparity estimation, which uses disparity estimates from coarse scales to select the populations used at finer scales and can effectively detect occlusions.

## 1 Introduction

Binocular disparity, the displacement between the image locations of an object between two eyes or cameras, is an important depth cue. Mammalian brains appear to represent the stimulus disparity using populations of disparity-tuned neurons in the visual cortex [1][2]. The binocular energy model is a first order model that explains the responses of individual disparity-tuned neurons [3]. In this model, the preferred disparity tuning of the neurons is determined by the phase and position shifts between the left and right monocular receptive fields (RFs).

Peak picking is a common disparity estimation strategy for these neurons([4]-[6]). In this strategy, the disparity estimates are computed by the preferred disparity of the neuron with the largest response among the neural population. Chen and Qian [4] have suggested that the peak location in a population of phase-tuned disparity energy neurons is a more reliable estimate than the peak location in a population of position-tuned neurons.

It is difficult to estimate disparity from a single phase-tuned neuron population because its range of preferred disparities is limited. Figure 1 shows the population response of phase-tuned neurons (vertical cross section) for different stimulus disparities. If the stimulus disparity is confined to the range of preferred disparities of this population, the peak location changes linearly with the stimulus disparity. Thus, we can estimate the disparity from the peak. However, in natural viewing condition, the stimulus disparity ranges over ten times larger than the range of the preferred disparities of the population [7]. The peak location no longer indicates the stimulus disparity, since the peaks still occur even when the stimulus disparity is outside the range of neurons' preferred disparities. The false peaks arise from two sources: the phase wrap-around due to the sinusoidal modulation in the

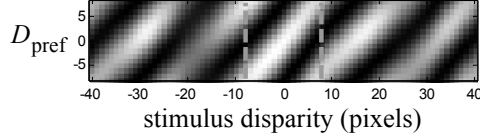

$D_{\text{pref}}$

stimulus disparity (pixels)

Fig. 1: Sample population responses of the phase-tuned disparity neurons for different disparities. This was generated by presenting the left image of the "Cones" stereogram shown in Figure 5a to both eyes but varying the disparity by keeping the left image fixed and shifting the right image. At each point, the image intensity represents the response of a disparity neuron tuned to a fixed preferred disparity (vertical axis) in response to a fixed stimulus disparity (horizontal axis). The dashed vertical lines indicate the stimulus disparities that fall within the range of preferred disparities of the population ($\pm 8$ pixels).

Gabor function modelling neuron's receptive field (RF) profile, or unmatching edges entering the neuron's RF [5].

Although a single population can cover a large disparity range, the large size of the required receptive fields results in very low resolution depth estimates. To address this problem, Chen and Qian [4] proposed a coarse-to-fine algorithm which refines the estimates computed from coarse scales using populations tuned to finer scales.

Here we present an alternative way to estimate the stimulus disparity using a biologically plausible confidence measure that indicates whether the stimulus disparity lies inside or outside the range of preferred disparities in a population of phase tuned neurons. We motivate this measure by examining the empirical statistics of the model neuron responses on natural images. Finally, we demonstrate the efficacy of using this measure to estimate the stimulus disparity. Our model generates better estimates than the coarse-to-fine approach [4], and can detect occlusions.

## 2 Features of the phase-tuned disparity population

In this section, we define different features of a population of phase-tuned neurons. These features will be used to define the confidence measure. Figure 2a illustrates the binocular disparity energy model of a phase-tuned neuron [3]. For simplicity, we assume 1D processing, which is equivalent to considering one orientation in the 2D case. The response of a binocular simple cell is modelled by summing of the outputs of linear monocular Gabor filters applied to both left and right images, followed by a positive or negative half squaring nonlinearity. The response of a binocular complex cell is the sum of the four simple cell responses.

Formally, we define the left and right retinal images by $U_l(x)$ and $U_r(x)$, where $x$ denotes the distance from the RF center. The disparity $d$ is the difference between the locations of corresponding points in the left and right images, i.e., an object that appears at point $x + d$ in the left image appears at point $x$ in the right image. Pairs of monocular responses are generated by integrating image intensities weighted by pairs of phase quadrature RF profiles, which are the real and imaginary parts of a complex-valued Gabor function ($j = \sqrt{-1}$):

$$h(x, \psi) = g(x)e^{j(\Omega x + \psi)} = g(x)\cos(\Omega x + \psi) + jg(x)\sin(\Omega x + \psi) \qquad (1)$$

where $\Omega$ and $\psi$ are the spatial frequency and the phase of the left and right monocular RFs, and $g(x)$ is a zero mean Gaussian with standard deviation $\sigma$, which is inversely proportional to the spatial frequency bandwidth. The spatial frequency and the standard deviation of the left and right RFs are identical, but the phases may differ ($\psi_l$ and $\psi_r$). We can compactly express the pairs of left and right monocular responses as the real and imaginary parts of $V_l(\psi_l) = V_l e^{j\psi_l}$ and $V_r(\psi_r) = V_r e^{j\psi_r}$, where with a slight abuse of notation, we define

$$V_l = \int g(x)e^{j\Omega x}U_l(x)dx \text{ and } V_r = \int g(x)e^{j\Omega x}U_r(x)dx \qquad (2)$$

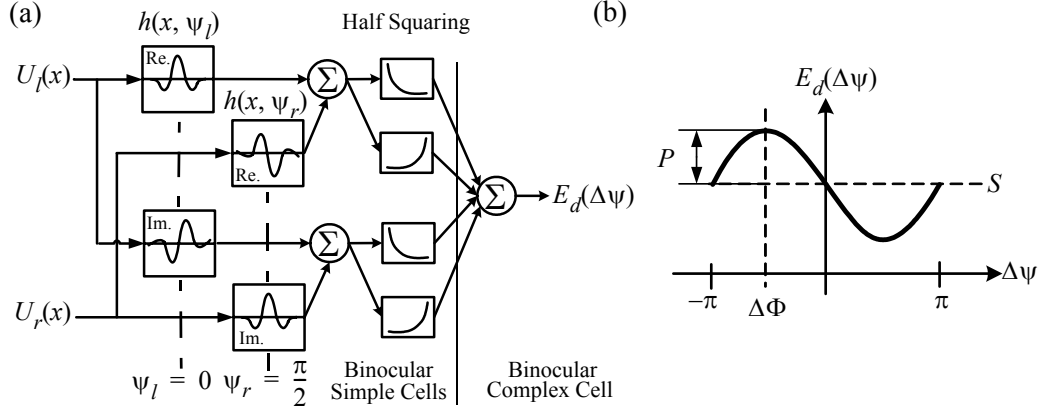

Fig. 2: (a) Binocular disparity energy model of a disparity neuron in the phase-shift mechanism. The phase-shift $\psi_r - \psi_l$ between the left and right monocular RFs determines the preferred disparity of the neuron. The neuron shown is tuned to a negative disparity of $-\pi/(2\Omega)$. (b) The population response of the phase-tuned neurons $E_d(\Delta\psi)$ centered at a retinal location with the phase-shifts $\Delta\psi \in [-\pi, \pi]$ can be characterized by three features $S, P$ and $\Delta\Phi$.

The response of the binocular complex cell (the disparity energy) is the squared modulus of the sum of the monocular responses:

$$E_d(\Delta\psi) = \left\| V_l e^{j\psi_l} + V_r e^{j\psi_r} \right\|^2 = \|V_l\|^2 + V_l V_r^* e^{-j\Delta\psi} + V_l^* V_r e^{j\Delta\psi} + \|V_r\|^2 \tag{3}$$

where the * superscript indicates the complex conjugation. The phase-shift between the right and left neurons $\Delta\psi = \psi_r - \psi_l$ controls the preferred disparity $D_{\text{pref}}(\Delta\psi) \approx -\Delta\psi/\Omega$ of the binocular complex cell [6].

If we fix the stimulus and allow $\Delta\psi$ to vary between $\pm\pi$, the function $E_d(\Delta\psi)$ in (3) describes the population response of phase-tuned neurons whose preferred disparities range between $-\pi/\Omega$ and $\pi/\Omega$. The population response can be completely specified by three features $S$, $P$ and $\Delta\Phi$ [4][5].

$$E_d(\Delta\psi) = S + P\cos(\Delta\Phi - \Delta\psi) \tag{4}$$

where

$$S = \|V_l\|^2 + \|V_r\|^2$$
$$P = 2\|V_l\|\|V_r\| = 2\|V_l V_r^*\| \tag{5}$$
$$\Delta\Phi = \Phi_l - \Phi_r = \arg(V_l V_r^*)$$

Figure 2b shows the graphical interpretation of these features. The feature $S$ is the average response across the population. The feature $P$ is the difference between the peak and average responses. Note that $S \geq P$, since $S - P = (\|V_l\| - \|V_r\|)^2 > 0$. The feature $\Delta\Phi$ is the peak location of the population response. Peak picking algorithms compute the estimates from the peak location, i.e. $d_{\text{est}} = -\Delta\Phi/\Omega$ [6].

## 3  Feature Analysis

In this section, we suggest a simple confidence measure that can be used to differentiate between two classes of stimulus disparities: DIN and DOUT corresponding to stimulus disparities inside ($|d| \leq \pi/\Omega$) and outside ($|d| > \pi/\Omega$) the range of preferred disparities in the population.

We find this confidence measure by analyzing the empirical joint densities of $S$ and the ratio $R = P/S$ conditioned on the two disparity classes. Considering $S$ and $R$ is equivalent to considering $S$ and $P$. We ignore $\Delta\Phi$. Intuitively, the peak location $\Delta\Phi$ will be less effective in distin-

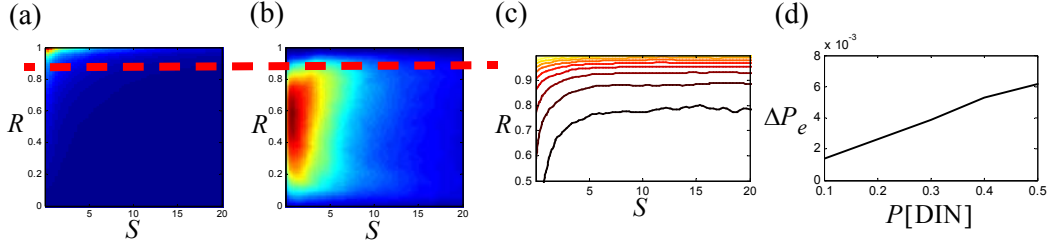

Fig. 3: The empirical joint density of $S$ and $R$ given (a) DIN and (b) DOUT. Red indicates large values. Blue indicates small values. (c) The optimal decision boundaries derived from the Bayes factor. (d) The change in total probability of error $\Delta P_e$ between using a flat boundary (thresholding $R$) versus the optimal boundary.

guishing between DIN and DOUT, since Figure 1 shows that the phase ranges between $-\pi$ and $\pi$ for both disparity classes. The ratio $R$ is bounded between $0$ and $1$, since $S \geq P$.

Because of the uncertainties in the natural scenes, the features $S$ and $R$ are random variables. In making a decision based on random features, Bayesian classifiers minimize the classification error. Bayesian classifiers compare the conditional probabilities of the two disparity classes (DIN and DOUT) given the observed feature values. The decision can be specified by thresholding the Bayes factor.

$$B_{S,R} = \frac{f_{S,R|C}(s,r|\mathrm{DIN})}{f_{S,R|C}(s,r|\mathrm{DOUT})} \underset{\mathrm{DOUT}}{\overset{\mathrm{DIN}}{\lessgtr}} T_{S,R} \tag{6}$$

where the threshold $T_{S,R}$ controls the location of the decision boundary in the feature space $\{S,R\}$ and depends upon the prior class probabilities $P[\mathrm{DIN}]$ and $P[\mathrm{DOUT}]$. The function $f_{S,R|C}(s,r|c)$ is the conditional density of the features given the class $c \in \{\mathrm{DIN}, \mathrm{DOUT}\}$.

To find the optimal decision boundary for the features $S$ and $R$, we estimated the joint class likelihood $f_{S,R|C}(s,r|c)$ from data obtained using the "Cones" and the "Teddy" stereograms from Middlebury College [8][9], shown in Figure 5a. The stereograms are rectified, so that the correspondences are located in the same horizontal scan-lines. Each image has 1500 x 1800 pixels. We constructed a population of phase-tuned neurons at each pixel. The disparity neurons had the same spatial frequency and standard deviation, and were selective to vertical orientations. The spatial frequency was $\Omega = 2\pi/16$ radians per pixel and the standard deviation in the horizontal direction was $\sigma = 6.78$ pixels, corresponding to a spatial bandwidth of 1.8 octaves. The standard deviation in the vertical direction was $2\sigma$. The range of the preferred disparities (DIN) of the population is between $\pm 8$ pixels. To reduce the variability in the classification, we also applied Gaussian spatial pooling with the standard deviation $0.5\sigma$ to the population [4][5]. The features $S$ and $R$ computed from population were separated into two classes (DIN and DOUT) according to the ground truth in Figure 5b.

Figure 3a-b show the empirically estimated joint conditional densities for the two disparity classes. They were computed by binning the features $S$ and $R$ with the bin sizes of 0.25 for $S$ and 0.01 for $R$. Given the disparity within the range of preferred disparities (DIN), the joint density concentrates at small $S$ and large $R$. For the out-of-range disparities (DOUT), the joint density shifts to both large $S$ and small $R$. Intuitively, a horizontal hyperplane, illustrated by the red dotted line in Figure 3a-b, is an appropriate decision boundary to separate the DIN and DOUT data. This indicates that the feature $R$ can be an indicator to distinguish between the in-range and out-of-range disparities. Mathematically, we can compute the optimal decision boundaries by applying different thresholds to the Bayes factor in (6). Figure 3c shows the boundaries. They are basically flat except at small $S$.

We also demonstrate the efficacy of thresholding $R$ instead of using the optimal decision boundaries to distinguish between in-range and out-of-range disparities. Given the prior class probability

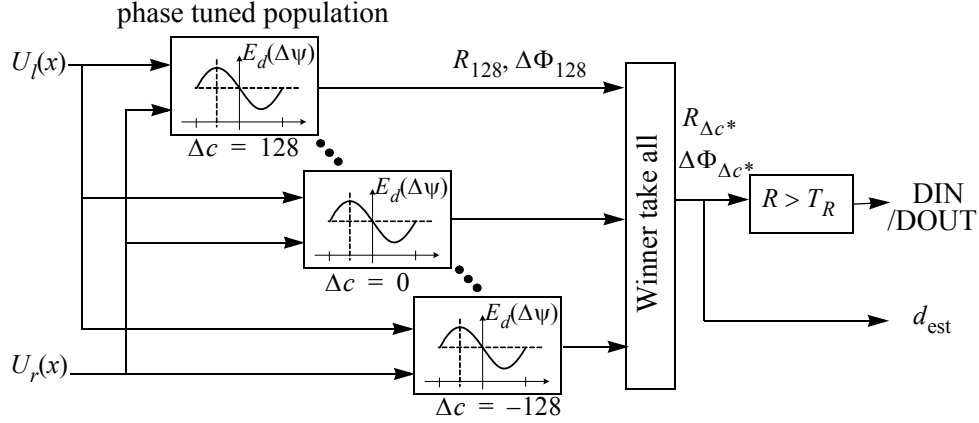

Fig. 4: Proposed disparity estimator with the validation of disparity estimates.

$P[\text{DIN}]$, we compute a threshold $c \in [0, 1]$ that minimizes the total probability of classification error:

$$P_e = P[\text{DIN}] \int_{R<c} f_{S,R|C}(s, r|\text{DIN}) + (1 - P[\text{DIN}]) \int_{R>c} f_{S,R|C}(s, r|\text{DOUT}) \qquad (7)$$

We then compare this total probability of error with the one computed using the optimal decision boundaries derived in (6). Figure 3d shows the deviation in the total probability of error between the two approaches versus $P[\text{DIN}]$. The deviation is small (on the order of $10^{-2}$) suggesting that thresholding $R$ results in similar performance as using the optimal decision boundaries. Thus, $R$ can be used as a confidence measure for distinguishing DIN and DOUT. Moreover, this measure can be computed by normalization, which is a common component in models for V1 neurons [11].

## 4 Hybrid position-phase model for disparity estimation with validation

Our analysis above shows that $R$ is a simple indicator to distinguish between in-range and out-of-range disparities. In this section, we describe a model that uses this feature to estimate the stimulus disparity with validation.

Figure 4 shows the proposed model, which consists of populations of hybrid tuned disparity neurons tuned to different phase-shifts $\Delta \psi$ and position-shifts $\Delta c$. For each population tuned to the same position-shift but different phase-shifts (phase-tuned population), we compute the ratio $R_{\Delta c} = P_{\Delta c} / S_{\Delta c}$. The average activation $S_{\Delta c}$ can be computed by pooling the responses of the entire phase-tuned neurons. The feature $P_{\Delta c}$ can be computed by subtracting the peak response $S_{\Delta c} + P_{\Delta c}$ of the phase tuned population with the average activation $S_{\Delta c}$. The features $R_{\Delta c}$ at different position-shifts are compared through a winner-take-all network to select the position-shift $\Delta c^*$ with the maximum $R_{\Delta c}$. The disparity estimate is further refined by the peak location $\Delta \Phi_{\Delta c^*}$ by

$$d_{\text{est}} = \Delta c^* - \frac{\Delta \Phi_{\Delta c^*}}{\Omega} \qquad (8)$$

In additional to estimate the stimulus disparity, we also validate the estimates by comparing $R_{\Delta c^*}$ with a threshold $T_R$. Instead of choosing a fixed threshold, we vary the threshold to show that the feature $R_{\Delta c}$ can be an occlusion detector.

### 4.1 Disparity estimation with confidence

We applied the proposed model to estimate the disparity of the "Cones" and the "Teddy" stereograms, shown in Figure 5a. The spatial frequency and the spatial standard deviation of the neurons

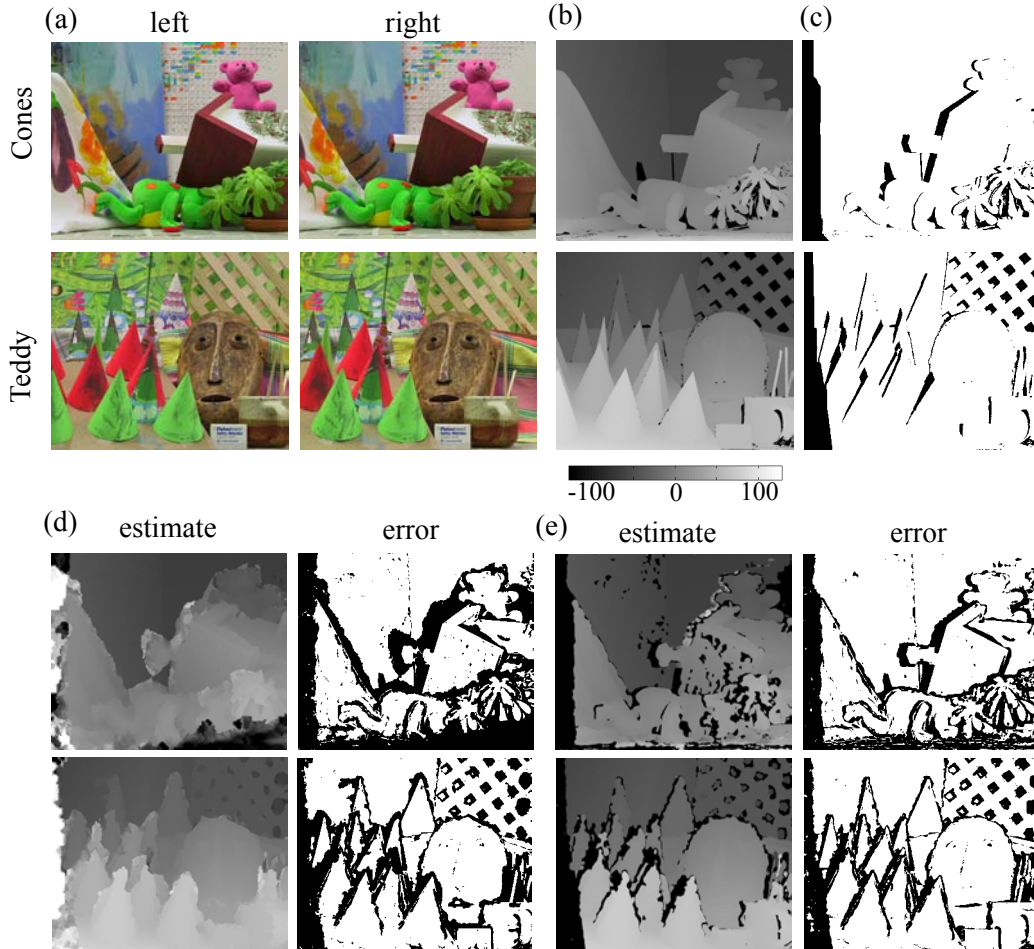

Fig. 5: (a) The two natural stereograms used to evaluate the model performance. (b) The ground truth disparity maps with respect to the left images, obtained by the structured light method. (c) The ground truth occlusion maps. (d) The disparity maps and the error maps computed by the coarse-to-fine approach. (e) The disparity maps and the error maps computed by the proposed model. The detected invalid estimates are labelled in black in the disparity maps.

were kept the same as the previous analysis. We also performed spatial pooling and orientation pooling to improve the estimation. For spatial pooling, we applied a circularly symmetric Gaussian function with standard deviation $\sigma$. For orientation pooling, we pooled the responses over five orientations ranging from 30 to 150 degrees. The range of the position-shifts for the populations was set to the largest disparity range, $\pm 128$ pixels, according to the ground truth.

We also implemented the coarse-to-fine model as described in [4] for comparison. In this model, an initial disparity estimate computed from a population of phase-tuned neurons at the coarsest scale is successively refined by the populations of phase-tuned neurons at the finer scales. By choosing the coarsest scale large enough, the disparity range covered by this method can be arbitrarily large. The coarsest and the finest scales had the Gabor periods of 512 and 16 pixels. The Gabor periods of the successive scales differed by a factor of $\sqrt{2}$. Neurons at the finest scale had the same RF parameters as our model. Same spatial pooling and orientation pooling were applied on each scale.

Figure 5d-e show the estimated disparity maps and the error maps of the two approaches. The error maps show the regions where the disparity estimates exceed 1 pixel of error in the disparity. Both models correctly recover the stimulus disparity at most locations with gradual disparity changes, but tend to make errors at the depth boundaries. However, the proposed model generates more accurate estimates. In the coarse-to-fine model, the percentage of pixels being incorrectly estimated is 36.3%, while our proposed model is only 27.8%.

The coarse-to-fine model tends to make errors around the depth boundaries. This arises because the assumption that the stimulus disparity is constant over the RF of the neuron is unlikely at very large scales. At boundaries, the coarse-to-fine model generates poor initial estimates, which cannot be corrected at the finer scales, because the actual stimulus disparities are outside the range considered at the finer scales.

On the other hand, the proposed model can not only estimate the stimulus disparity, but also can validate the estimates. In general, the responses of neurons selective to different position disparities are not comparable, since they depend upon image contrast which varies at different spatial locations. However, the feature $R$, which is computed by normalizing the response peak by the average response, eliminates such dependency. Moreover, the invalid regions detected (the black regions on the disparity maps) are in excellent agreement with the error labels.

## 4.2 Occlusion detection

In addition to validating the disparity estimates, the feature $R$ can also be used to detect occlusion. Occlusion is one of the challenging problems in stereo vision. Occlusion occurs near the depth discontinuities where there is no correspondence between the left and right images. The disparity in the occlusion regions is undefined. The occlusion regions for these stereograms are shown in Figure 5c.

There are three possibilities for image pixels that are labelled as out of range (DOUT). They are occluded pixels, pixels with valid disparities that are incorrectly estimated, and pixels with valid disparity that are correctly estimated. Figure 6a shows the percentages of DOUT pixels that fall into each possibility as the threshold $T_R$ applied to $R$ varies, e.g.,

$$P1(\text{occluded}) = \frac{\text{\# of occluded pixels in DOUT}}{\text{total \# of pixels in DOUT}} \times 100\% \qquad (9)$$

These percentages sum to unity for any thresholds $T_R$. For small thresholds, the detector mainly identifies the occlusion regions. As the threshold increases, the detector also begins to detect incorrect disparity estimates. Figure 6b shows the percentages of pixels in each possibility that are classified as DOUT as a function of $T_R$, e.g.,

$$P2(\text{occluded}) = \frac{\text{\# of occluded pixels in DOUT}}{\text{\# of occluded pixels in image}} \times 100\% \qquad (10)$$

For a large threshold ($T_R$ close to unity), all estimates are labelled as DOUT, so the three percentages approach 100%. The proposed detector is effective in identifying occlusion. At the threshold $T_R = 0.3$, it identifies ~70% of the occluded pixels, ~20% of the pixels with incorrect estimates with only ~10% misclassification.

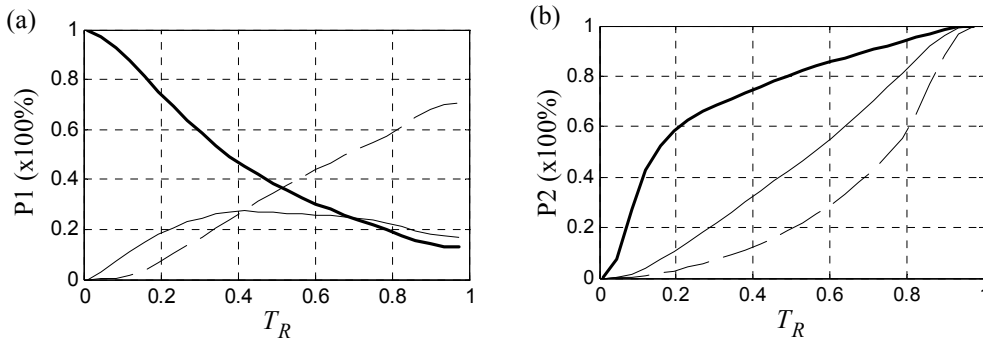

Fig. 6: The percentages of occluded pixels (thick), pixels with incorrect disparity estimates (thin) and pixels with correct estimates (dotted) identified as DOUT. (a) Percentages as a fraction of total number of DOUT pixels. (b) Percentages as a fraction of number of pixels of each type.

# 5  Discussion

In this paper, we have proposed an algorithm to estimate stimulus disparities based on a confidence measure computed from population of hybrid tuned disparity neurons. Although there have been previously proposed models that estimate the stimulus disparity from populations of hybrid tuned neurons [4][10], our model is the first that also provides a confidence measure for these estimates. Our analysis suggests that pixels with low confidence are likely to be in occluded regions. The detection of occlusion, an important problem in stereo vision, was not addressed in these previous approaches.

The confidence measure used in the proposed algorithm can be computed using normalization, which has been used to model the responses of V1 neurons [11]. Previous work has emphasized the role of normalization in reducing the effect of image contrast or in ensuring that the neural responses tuned to different stimulus dimensions are comparable [12]. Our results show that, in addition to these roles, normalization also serves to make the magnitude of the neural responses more representative of the confidence in validating the hypothesis that the input disparity is close to the neurons preferred disparity. The classification performance using this normalized feature is close to that using the statistical optimal boundaries.

Aggregating the neural responses over locations, orientations and scales is a common technique to improve the estimation performance. For the consistency with the coarse-to-fine approach, our algorithm also applies spatial and orientation pooling before computing the confidence. An interesting question, which we are now investigating, is whether individual confidence measures computed from different locations or orientations can be combined systematically.

## Acknowledgements

This work was supported in part by the Hong Kong Research Grants Council under Grant 619205.

## References

[1]    H. B. Barlow, C. Blakemore, and J. D. Pettigrew. The neural mechanism of binocular depth discrimination. *Journal of Neurophysiology,* vol. 193(2), 327-342, 1967.

[2]    G. F. Poggio, B. C. Motter, S. Squatrito, and Y. Trotter. Responses of neurons in visual cortex (V1 and V2) of the alert macaque to dynamic random-dot stereograms. *Vision Research,* vol. 25, 397-406, 1985.

[3]    I. Ohzawa, G. C. Deangelis, and R. D. Freeman. Stereoscopic depth discrimination in the visual cortex: neurons ideally suited as disparity detectors. *Science,* vol. 249, 1037-1041, 1990.

[4]    Y. Chen and N. Qian. A Coarse-to-Fine Disparity Energy Model with Both Phase-Shift and Position-Shift Receptive Field Mechanisms. *Neural Computation,* vol. 16, 1545-1577, 2004.

[5]    D. J. Fleet, H. Wagner and D. J. Heeger. Neural encoding of binocular disparity: energy models, position shifts and phase shifts. *Vision Research,* 1996, vol. 36, 1839-1857.

[6]    N. Qian, and Y. Zhu. Physiological computation of binocular disparity. *Vision Research,* vol. 37, 1811-1827, 1997.

[7]    S. J. D. Prince, B. G. Cumming, and A. J. Parker. Range and Mechanism of Encoding of Horizontal Disparity in Macaque V1. *Journal of Neurophysiology,* vol. 87, 209-221, 2002.

[8]    D. Scharstein and R. Szeliski. A Taxonomy and Evaluation of Dense Two-Frame Stereo Correspondence Algorithms. *International Journal of Computer Vision,* vol. 47(1/2/3), 7-42, 2002.

[9]    D. Scharstein and R. Szeliski. High-accuracy stereo depth maps using structured light. *IEEE Conference on Computer Vision and Pattern Recognition,* vol. 1, 195-202, 2003.

[10]   J. C. A. Read and B. G. Cumming. Sensors for impossible stimuli may solve the stereo correspondence problem. *Nature Neuroscience,* vol. 10, 1322-1328, 2007.

[11]   D. J. Heeger. Normalization of cell responses in cat striate cortex. *Visual Neuroscience,* vol. 9, 181-198, 1992.

[12]   S. R. Lehky and T. J. Sejnowski. Neural model of stereoacuity and depth interpolation based on a distributed representation of stereo disparity. *Journal of Neuroscience,* vol. 10, 2281-2299, 1990.
